# Training sparse natural image models with a fast Gibbs sampler of an extended state space

**Lucas Theis**
Werner Reichardt Centre
for Integrative Neuroscience
lucas@bethgelab.org

**Jascha Sohl-Dickstein**
Redwood Center
for Theoretical Neuroscience
jascha@berkeley.edu

**Matthias Bethge**
Werner Reichardt Centre
for Integrative Neuroscience
matthias@bethgelab.org

## Abstract

We present a new learning strategy based on an efficient blocked Gibbs sampler for sparse overcomplete linear models. Particular emphasis is placed on statistical image modeling, where overcomplete models have played an important role in discovering sparse representations. Our Gibbs sampler is faster than general purpose sampling schemes while also requiring no tuning as it is free of parameters. Using the Gibbs sampler and a *persistent* variant of expectation maximization, we are able to extract highly sparse distributions over latent sources from data. When applied to natural images, our algorithm learns source distributions which resemble *spike-and-slab* distributions. We evaluate the likelihood and quantitatively compare the performance of the overcomplete linear model to its complete counterpart as well as a product of experts model, which represents another overcomplete generalization of the complete linear model. In contrast to previous claims, we find that overcomplete representations lead to significant improvements, but that the overcomplete linear model still underperforms other models.

## 1 Introduction

Here we study learning and inference in the overcomplete linear model given by

$$x = As, \quad p(s) = \prod_i f_i(s_i), \tag{1}$$

where $A \in \mathbb{R}^{M \times N}$, $N \geq M$, and each marginal source distribution $f_i$ may depend on additional parameters. Our goal is to find parameters which maximize the model's log-likelihood, $\log p(\mathbf{x})$, for a given set of observations $\mathbf{x}$.

Most of the literature on overcomplete linear models assumes observations corrupted by additive Gaussian noise, that is, $x = As + \varepsilon$ for a Gaussian distributed random variable $\varepsilon$. Note that this is a special case of the model discussed here, as we can always represent this noise by making some of the sources Gaussian.

When the observations are image patches, the source distributions $f_i(s_i)$ are typically assumed to be sparse or leptokurtotic [e.g., 2, 20, 28]. Examples include the Laplace distribution, the Cauchy distribution, and Student's *t*-distribution. A large family of leptokurtotic distributions which also contains

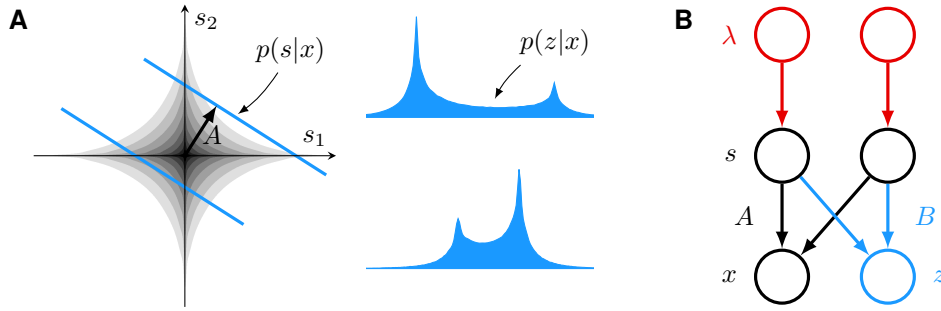

Figure 1: **A:** In the noiseless overcomplete linear model, the posterior distribution over hidden sources $s$ lives on a linear subspace. The two parallel lines indicate two different subspaces for different values of $x$. For sparse source distributions, the posterior will generally be heavy-tailed and multimodal, as can be seen on the right. **B:** A graphical model representation of the overcomplete linear model extended by two sets of auxiliary variables (Equation 2 and 3). We perform blocked Gibbs sampling between $\lambda$ and $z$ to sample from the posterior distribution over all latent variables given an observation $x$. For a given $\lambda$, the posterior over $z$ becomes Gaussian while for given $z$, the posterior over $\lambda$ becomes factorial and is thus easy to sample from.

the aforementioned distributions as a special case is formed by *Gaussian scale mixtures* (GSMs),

$$f_i(s_i) = \int_0^\infty g_i(\lambda_i)\mathcal{N}(s_i; 0, \lambda_i^{-1})\, d\lambda_i, \tag{2}$$

where $g_i(\lambda_i)$ is a univariate density over precisions $\lambda_i$. In the following, we will concentrate on linear models whose marginal source distributions can be represented as GSMs. For a detailed description of the representational power of GSMs, see Andrews and Mallows' paper [1].

Despite the apparent simplicity of the linear model, inference over the latent variables is computationally hard except for a few special cases such as when all sources are Gaussian distributed. In particular, the posterior distribution over sources $p(s \mid x)$ is constrained to a linear subspace and can have multiple modes with heavy tails (Figure 1A).

Inference can be simplified by assuming additive Gaussian noise, constraining the source distributions to be log-concave or making crude approximations to the posterior. Here, however, we would like to exhaust the full potential of the linear model. On this account, we use Markov chain Monte Carlo (MCMC) methods to obtain samples with which we represent the posterior distribution. While computationally more demanding than many other methods, this allows us, at least in principle, to approximate the posterior to arbitrary precision.

Other approximations often introduce strong biases and preclude learning of meaningful source distributions. Using MCMC, on the other hand, we can study the model's optimal sparseness and overcompleteness level in a more objective fashion as well as evaluate the model's log-likelihood.

However, multiple modes and heavy tails also pose challenges to MCMC methods. General purpose methods are therefore likely to be slow. In the following, we will describe an efficient blocked Gibbs sampler which exploits the specific structure of the sparse linear model.

## 2  Sampling and inference

In this section, we first review the *nullspace sampling algorithm* of Chen and Wu [4], which solves the problem of sampling from a linear subspace in the noiseless case of the overcomplete linear model. We then introduce an additional set of auxiliary variables which leads to an efficient blocked Gibbs sampler.

## 2.1 Nullspace sampling

The basic idea behind the nullspace sampling algorithm is to extend the overcomplete linear model by an additional set of variables $z$ which essentially makes it complete (Figure 1B),

$$\left[\begin{array}{c} x \\ z \end{array}\right] = \left[\begin{array}{c} A \\ B \end{array}\right] s, \tag{3}$$

where $B \in \mathbb{R}^{(N-M) \times N}$ and square brackets denote concatenation. If in addition to our observation $x$ we knew the unobserved variables $z$, we could perform inference as in the complete case by simply solving the above linear system, provided the concatenation of $A$ and $B$ is invertible. If the rows of $A$ and $B$ are orthogonal, $AB^\top = 0$, or, in other words, $B$ spans the nullspace of $A$, we have

$$s = A^+ x + B^+ z, \tag{4}$$

where $A^+$ and $B^+$ are the pseudoinverses [24] of $A$ and $B$, respectively. The marginal distributions over $x$ and $s$ do not depend on our choice of $B$, which means we can choose $B$ freely. An orthogonal basis spanning the nullspace of $A$ can be obtained from $A$'s singular value decomposition [4].

Making use of Equation 4, we can equally well try to obtain samples from the posterior $p(z \mid x)$ instead of $p(s \mid x)$. In contrast to the latter, this distribution has full support and is not restricted to just a linear subspace,

$$p(z \mid x) \propto p(z, x) \propto p(s) = \prod_i f_i(w_i^\top x + v_i^\top z), \tag{5}$$

where $w_i^\top$ and $v_i^\top$ are the $i$-th rows of $A^+$ and $B^+$, respectively. Chen and Wu [4] used Metropolis-adjusted Langevin (MALA) sampling [25] to sample from $p(z \mid x)$.

## 2.2 Blocked Gibbs sampling

The fact that the marginals $f_i(s_i)$ are expressed as Gaussian mixtures (Equation 2) can be used to derive an efficient blocked Gibbs sampler. The Gibbs sampler alternately samples nullspace representations $z$ and precisions of the source marginals $\lambda$. The key observation here is that given the precisions $\lambda$, the distribution over $x$ and $z$ becomes Gaussian which makes sampling from the posterior distribution tractable.

A similar idea was pursued by Olshausen and Millman [21], who modeled the source distributions with mixtures of Gaussians and conditionally Gibbs sampled precisions one by one. However, a change in one of the precision variables entails larger computational costs, so that this algorithm is most efficient if only few Gaussians are used and the probability of changing precisions is small. In contrast, here we update all precision variables in parallel by conditioning on the nullspace representation $z$. This makes it feasible to use a large or even infinite number of precisions.

Conditioned on a data point $x$ and a corresponding nullspace representation $z$, the distribution over precisions $\lambda$ becomes factorial,

$$p(\lambda \mid x, z) = p(\lambda \mid s) \propto p(s \mid \lambda)p(\lambda) = \prod_i \mathcal{N}(s_i; 0, \lambda_i^{-1})g_i(\lambda_i), \tag{6}$$

where we have used the fact that we can perfectly recover the sources given $x$ and $z$ (Equation 4). Using a finite number of precisions $\vartheta_{ik}$ with prior probabilities $\pi_{ik}$, for example, the posterior probability of $\lambda_i$ being $\vartheta_{ij}$ becomes

$$p(\lambda_i = \vartheta_{ij} \mid x, z) = \frac{\mathcal{N}(s_i; 0, \vartheta_{ij}^{-1})\pi_{ij}}{\sum_k \mathcal{N}(s_i; 0, \vartheta_{ik}^{-1})\pi_{ik}}. \tag{7}$$

Conditioned on $\lambda$, $s$ is Gaussian distributed with diagonal covariance $\Lambda^{-1} = \mathrm{diag}(\lambda^{-1})$. As a linear transformation of $s$, the distribution over $x$ and $z$ is also Gaussian with covariance

$$\Sigma = \left[\begin{array}{cc} A\Lambda^{-1}A^\top & A\Lambda^{-1}B^\top \\ B\Lambda^{-1}A^\top & B\Lambda^{-1}B^\top \end{array}\right] = \left[\begin{array}{cc} \Sigma_{xx} & \Sigma_{xz} \\ \Sigma_{xz}^\top & \Sigma_{zz} \end{array}\right]. \tag{8}$$

Using standard Gaussian identities, we obtain

$$p(z \mid x, \lambda) = \mathcal{N}(z; \mu_{z|x}, \Sigma_{z|x}), \tag{9}$$

where $\mu_{z|x} = \Sigma_{xz}^{\top}\Sigma_{xx}^{-1}x$ and $\Sigma_{z|x} = \Sigma_{zz} - \Sigma_{xz}^{\top}\Sigma_{xx}^{-1}\Sigma_{xz}$. We use the following computationally efficient method to conditionally sample Gaussian distributions [8, 14]:

$$\left[ \begin{array}{c} x' \\ z' \end{array} \right] \sim \mathcal{N}(0, \Sigma), \quad z = z' + \Sigma_{xz}^{\top}\Sigma_{xx}^{-1}(x - x'). \tag{10}$$

It can easily be shown that $z$ has the desired distribution of Equation 9. Together, equations 7 and 9 implement a rapidly mixing blocked Gibbs sampler. However, the computational cost of solving Equation 10 is larger than for a single Markov step in other sampling methods such as MALA. We empirically show in the results section that for natural image patches the benefits of blocked Gibbs sampling outweigh its computational costs.

A closely related sampling algorithm was proposed by Park and Casella [23] for implementing Bayesian inference in the linear regression model with Laplace prior. The main differences here are that we also consider the noiseless case by exploiting the nullspace representation, that instead of using a fixed Laplace prior we will use the sampler to learn the distribution over source variables, and that we apply the algorithm in the context of image modeling. Related ideas were also discussed by Papandreou and Yuille [22], Schmidt et al. [27], and others.

## 3   Learning

In the following, we describe a learning strategy for the overcomplete linear model based on the idea of *persistent Markov chains* [26, 32, 36], which already has led to improved learning strategies for a number of different models [e.g., 6, 12, 29, 32].

Following Girolami [11] and others, we use expectation maximization (EM) [7] to maximize the likelihood of the overcomplete linear model. Instead of a variational approximation, here we use the blocked Gibbs sampler to sample a hidden state $z$ for every data point $x$ in the E-step. Each M-step then reduces to maximum likelihood learning as in the complete case, for which many algorithms are available. Due to the sampling step, this variant of EM is known as Monte Carlo EM [34].

Despite our efforts to make sampling efficient, running the Markov chain till convergence can still be a costly operation due to the generally large number of data points and high dimensionality of posterior samples. To further reduce computational costs, we developed a learning strategy which makes use of persistent Markov chains and only requires a few sampling steps in every iteration.

Instead of starting the Markov chain anew in every iteration, we initialize the Markov chain with the samples of the previous iteration. This approach is based on the following intuition. First, if the model changes only slightly, the posterior will change only slightly. As a result, the samples from the previous iteration will provide a good initialization and fewer updates of the Markov chain will be sufficient to reach convergence. Second, if updating the Markov chain has only a small effect on the posterior samples $\mathbf{z}$, also the distribution of the complete data $(\mathbf{x}, \mathbf{z})$ will change very little. Thus, the optimal parameters of the previous M-step will be close to optimal in the current M-step. This causes an inefficient Markov chain to automatically slow down the learning process, so that the posterior samples will always be close to the stationary distribution.

Even updating the Markov chain only once results in a valid EM strategy, which can be seen as follows. EM can be viewed as alternately optimizing a lower bound to the log-likelihood with respect to model parameters $\theta$ and an approximating posterior distribution $q$ [18]:

$$F[q, \theta] = \log p(\mathbf{x}; \theta) - D_{\mathrm{KL}}\left[q(\mathbf{z} \mid \mathbf{x}) \,||\, p(\mathbf{z} \mid \mathbf{x}, \theta)\right]. \tag{11}$$

Each M-step increases $F$ for fixed $q$ while each E-step increases $F$ for fixed $\theta$. This is repeated until a local optimum is reached. Importantly, local maxima of $F$ are also local maxima of the log-likelihood, $\log p(\mathbf{x}; \theta)$.

Interestingly, improving the lower bound $F$ with respect to $q$ can be accomplished by driving the Markov chain with our Gibbs sampler or some other transition operator [26]. This can be seen

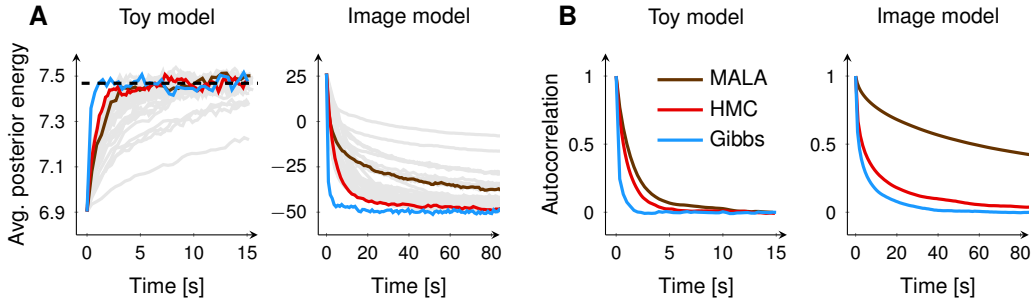

Figure 2: **A:** The average energy of posterior samples for different sampling methods after deterministic initialization. Depending on the initialization, the average energy can be initially too low or too high. Gray lines correspond to different hyperparameter choices for the HMC sampler, red and brown lines indicate the manually picked best performing HMC and MALA samplers. The dashed line represents an unbiased estimate of the true average posterior energy. **B:** Autocorrelation functions for Gibbs sampling and the best HMC and MALA samplers.

by using the fact that application of a transition operator $T$ to any distribution cannot increase its Kullback-Leibler (KL) divergence to a stationary distribution [5, 15]:

$$D_{\text{KL}}\left[Tq(\mathbf{z} \mid \mathbf{x}) \, \| \, p(\mathbf{z} \mid \mathbf{x}, \theta)\right] \leq D_{\text{KL}}\left[q(\mathbf{z} \mid \mathbf{x}) \, \| \, p(\mathbf{z} \mid \mathbf{x}, \theta)\right], \tag{12}$$

where $Tq(z \mid x) = \int q(z_0 \mid x)T(z \mid z_0, x)\, dz_0$ and $T(z \mid z_0, x)$ is the probability density of making a transition from $z_0$ to $z$. Hence, each Gibbs update of the hidden states implicitly increases $F$. In practice, of course, we only have access to samples from $Tq$ and will never compute it explicitly.

This shows that the algorithm converges provided the log-likelihood is bounded. This stands in contrast to other contexts where persistent Markov chains have been successful but training can diverge [10]. To guarantee not only convergence but convergence to a local optimum of $F$, we would also have to prove $D_{\text{KL}}\left[T^n q(\mathbf{z} \mid \mathbf{x}) \, \| \, p(\mathbf{z} \mid \mathbf{x}, \theta)\right] \to 0$ for $n \to \infty$. Unfortunately, most results on MCMC convergence deal with convergence in *total variation*, which is weaker than convergence in KL divergence.

## 4  Results

We trained several linear models on log-transformed, centered and symmetrically whitened image patches extracted from van Hateren's dataset of natural images [33]. We explicitly modeled the DC component of the whitened image patches using a mixture of Gaussians and constrained the remaining components of the linear basis to be orthogonal to the DC component.

For faster convergence, we initialized the linear basis with the sparse coding algorithm of Olshausen and Field [19], which corresponds to learning with MAP inference and fixed marginal source distributions. After initialization, we optimized the basis using L-BFGS [3] during each M-step and updated the representation of the posterior using 2 steps of Gibbs sampling in each E-step. To represent the source marginals, we used finite GSMs (Equation 8) with 10 precisions $\vartheta_{ij}$ each and equal prior weights, that is, $\pi_{ij} = 0.1$. The source marginals were initialized by fitting them to samples from the Laplace distribution and later optimized using 10 iterations of standard EM at the beginning of each M-step.

### 4.1  Performance of the blocked Gibbs sampler

We compared the sampling performance of our Gibbs sampler to MALA sampling—as used by Chen and Wu [4]—as well as HMC sampling [9], which is a generalization of MALA. The HMC sampler has two parameters: a step width and a number of so called leap frog steps. In addition, we slightly randomized the step width to avoid problems with periodicity [17], which added an additional parameter to control the degree of randomization. After manually determining a reasonable range for the parameters of HMC, we picked 40 parameter sets for each model to test against our Gibbs sampler.

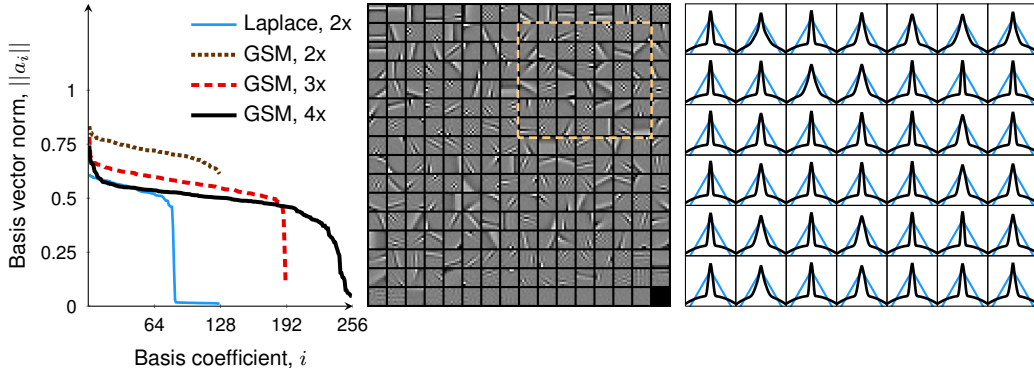

Figure 3: We trained models with up to four times overcomplete representations using either Laplace marginals or GSM marginals. A four times overcomplete basis set is shown in the center. Basis vectors were normalized so that the corresponding source distributions had unit variance. The left plot shows the norms of the learned basis vectors. With fixed Laplace marginals, the algorithm produces a basis which is barely overcomplete. However, with GSM marginals the model learns bases which are at least three times overcomplete. The right panel shows log-densities of the source distributions corresponding to basis vectors inside the dashed rectangle. For reference, each plot also contains a Laplace distribution of equal variance.

The algorithms were tested on one toy model and one two times overcomplete model trained on $8 \times 8$ image patches. The toy model employed 1 visible unit and 3 hidden units with exponential power distributions whose exponents were 0.5. The entries of its basis matrix were randomly drawn from a Gaussian distribution with mean 1 and standard deviation 0.2.

Figure 2 shows trace plots and autocorrelation functions for the different sampling methods. The trace plots were generated by measuring the negative log-density (or *energy*) of posterior samples for a fixed set of visible states over time, $-\log p(x, z_t)$, and averaging over data points. Autocorrelation functions were estimated from single Markov chain runs of equal duration for each sampler and data point. All Markov chains were initialized using 100 burn-in steps of Gibbs sampling, independent of the sampler used to generate the autocorrelation functions. Finally, we averaged several autocorrelation functions corresponding to different data points (see Supplementary Section 1 for more information).

For both models we observed faster convergence with Gibbs sampling than with the best MALA or HMC samplers (Figure 2). The image model in particular benefited from replacing MALA by HMC. Still, even the best HMC sampler produced more correlated samples than the blocked Gibbs sampler. While the best HMC sampler reached an autocorrelation of 0.05 after about 64 seconds, it took only about 26 seconds with the blocked Gibbs sampler (right-hand side of Figure 2B).

All tests were performed on a single core of an AMD Opteron 6174 machine with 2.20 GHz and implementations written in Python and NumPy.

## 4.2 Sparsity and overcompleteness

Berkes et al. [2] found that even for very sparse choices of the Student-t prior, the representations learned by the linear model are barely overcomplete if a variational approximation to the posterior is used. Similar results and even undercomplete representations were obtained by Seeger [28] with the Laplace prior. The results of these studies suggest that the optimal basis set is not very overcomplete. On the other hand, basis sets obtained with other, often more crude approximations are often highly overcomplete. In the following, we revisit the question of optimal overcompletness and support our findings with quantitative measurements.

Consistent with the study of Seeger [28], if we fix the source distributions to be Laplacian, our algorithm learns representations which are only slightly overcomplete (Figure 3). However, much more overcomplete representations were obtained when the source distributions were learned from the data. This is in line with the results of Olshausen and Millman [21], who used mixtures of two

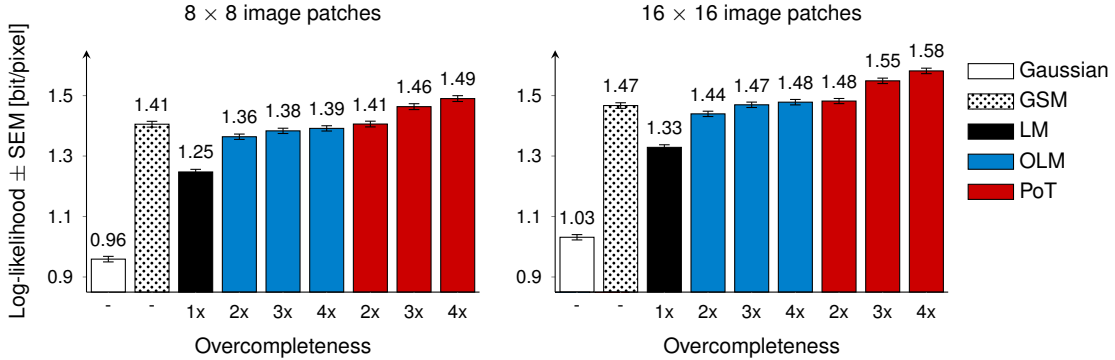

Figure 4: A comparison of different models for natural image patches. While using overcomplete representations (OLM) yields substantial improvements over the complete linear model (LM), it still cannot compete with other models of natural image patches. GSM here refers to a single multivariate Gaussian scale mixture, that is, an elliptically contoured distribution with very few parameters (see Supplementary Section 3). Log-likelihoods are reported for non-whitened image patches. Average log-likelihood and standard error of the mean (SEM) were calculated from log-probabilities of 10000 test data points.

and three Gaussians as source distributions and obtained two times overcomplete representations for $8 \times 8$ image patches.

Figure 3 suggests that with GSMs as source distributions, the model can make use of three and up to four times overcomplete representations. Our quantitative evaluations confirmed a substantial improvement of the two-times overcomplete model over the complete model. Beyond this, however, the improvements quickly become negligible (Figure 4).

The source distributions discovered by our algorithm were extremely sparse and resembled *spike-and-slab* distributions, generating mostly values close to zero with the occasional outlier. Source distributions of low-frequency components generally had narrower peaks than those of high-frequency components (Figure 3).

### 4.3 Model comparison

To compare the performance of the overcomplete linear model to the complete linear model and other image models, we would like to evaluate the overcomplete linear models' log-likelihood on a test set of images. However, to do this, we would have to integrate out all hidden units, which we cannot do analytically. One way to nevertheless obtain an unbiased estimate of $p(x)$ is by introducing a tractable distribution as follows:

$$p(x) = \int p(x, z) \, dz = \int q(z \mid x) \frac{p(x, z)}{q(z \mid x)} \, dz. \tag{13}$$

We can then estimate the above integral by sampling states $z_n$ from $q(z \mid x)$ and averaging over $p(x, z_n)/q(z_n \mid x)$, a technique called *importance sampling*. The closer $q(z \mid x)$ is to $p(z \mid x)$, the more efficient the estimator will be.

A procedure for constructing distributions $q(z \mid x)$ from transition operators such as our Gibbs sampling operator is *annealed importance sampling* (AIS) [16]. AIS starts with a simple and tractable distribution and successively brings it closer to $p(z \mid x)$. The computational and statistical efficiency of the estimator depends on the efficieny of the transition operator. Here, we used our Gibbs sampler and constructed intermediate distributions by interpolating between a Gaussian distribution and the overcomplete linear model. For the four-times overcomplete model, we used 300 intermediate distributions and 300 importance samples to estimate the density of each data point.

We find that the overcomplete linear model is still worse than, for example, a single multivariate GSM with separately modeled DC component (Figure 4; see also Supplementary Section 3).

An alternative overcomplete generalization of the complete linear model is the family of *products of experts* (PoE) [13]. Instead of introducing additional source variables, a PoE can have more factors than visible units,

$$s = Wx, \quad p(x) \propto \prod_i f_i(s_i), \tag{14}$$

where $W \in \mathbb{R}^{N \times M}$ and each factor is also called an *expert*. For $N = M$, the PoE is equivalent to the linear model (Equation 1). In contrast to the overcomplete linear model, the prior over hidden sources $s$ here is in general not factorial.

A popular choice of PoE in the context of natural images is the *product of Student-t* (PoT) distributions, in which experts have the form $f_i(s_i) = (1 + s_i^2)^{-\alpha_i}$ [35]. To train the PoT, we used a persistent variant of *minimum probability flow* learning [29, 31]. We used AIS in combination with HMC to evaluate each PoT model [30]. We find that the PoT is better suited for modeling the statistics of natural images and takes better advantage of overcomplete representations (Figure 4).

While both the estimator for the PoT and the estimator for the overcomplete linear model are consistent, the former tends to overestimate and the latter tends to underestimate the average log-likelihood. It is thus crucial to test convergence of both estimates if any meaningful comparison is to be made (see Supplementary Section 2).

## 5 Discussion

We have shown how to efficiently perform inference, training and evaluation in the sparse overcomplete linear model. While general purpose sampling algorithms such as MALA or HMC have the advantage of being more widely applicable, we showed that blocked Gibbs sampling can be much faster when the source distributions are sparse, as for natural images.

Another advantage of our sampler is that it is parameter free. Choosing suboptimal parameters for the HMC sampler can lead to extremely poor performance. Which parameters are optimal can change from data point to data point and over time as the model is trained. Furthermore, monitoring the convergence of the Markov chains can be problematic [28]. We showed that by training a model with a persistent variant of Monte Carlo EM, even the number of sampling steps performed in each E-step becomes much less crucial for the success of training.

Optimizing and evaluating the likelihood of overcomplete linear models is a challenging problem. To our knowledge, our study is the first to show a clear advantage of the overcomplete linear model over its complete counterpart on natural images. At the same time, we demonstrated that with the assumptions of a factorial prior, the overcomplete linear model underperforms other generalizations of the complete linear model. Yet it is easy to see how our algorithm could be extended to other, much better performing models. For instance, models in which multiple sources are modeled jointly by a multivariate GSM, or bilinear models with two sets of latent variables.

Code for training and evaluating overcomplete linear models is available at

$$\texttt{http://bethgelab.org/code/theis2012d/.}$$

**Acknowledgments**

The authors would like to thank Bruno Olshausen, Nicolas Heess and George Papandreou for helpful comments. This study was financially supported by the Bernstein award (BMBF; FKZ: 01GQ0601), the German Research Foundation (DFG; priority program 1527, BE 3848/2-1), and a DFG-NSF collaboration grant (TO 409/8-1).

**References**

[1] D. F. Andrews and C. L. Mallows. Scale mixtures of normal distributions. *Journal of the Royal Statistical Society, Series B*, 36(1):99–102, 1974.

[2] P. Berkes, R. Turner, and M. Sahani. On sparsity and overcompleteness in image models. *Advances in Neural Information Processing Systems*, 20, 2008.

[3] R. H. Byrd, P. Lu, and J. Nocedal. A limited memory algorithm for bound constrained optimization. *SIAM Journal on Scientific and Statistical Computing*, 16(5):1190–1208, 1995.

[4] R.-B. Chen and Y. N. Wu. A null space method for over-complete blind source separation. *Computational Statistics & Data Analysis*, 51(12):5519–5536, 2007.

[5] T. Cover and J. Thomas. *Elements of Information Theory*. Wiley, 1991.

[6] B. J. Culpepper, J. Sohl-Dickstein, and B. A. Olshausen. Building a better probabilistic model of images by factorization. *Proceedings of the International Conference on Computer Vision*, 13, 2011.

[7] A. P. Dempster, N. M. Laird, and D. B. Rubin. Maximum likelihood from incomplete data via the EM algorithm. *Journal of the Royal Statistical Society, Series B*, 39(1):1–38, 1977.

[8] A. Doucet. A note on efficient conditional simulation of Gaussian distributions, 2010.

[9] S. Duane, A. D. Kennedy, B. J. Pendleton, and D. Roweth. Hybrid Monte Carlo. *Physics Letters B*, 195 (2):216–222, 1987.

[10] A. Fischer and C. Igel. Empirical analysis of the divergence of Gibbs sampling based learning algorithms for restricted Boltzmann machines. *Proceedings of the 20th International Conference on Artificial Neural Networks*, 2010.

[11] M. Girolami. A variational method for learning sparse and overcomplete representations. *Neural Computation*, 13(11):2517–2532, 2001.

[12] N. Heess, N. Le Roux, and J. Winn. Weakly supervised learning of foreground-background segmentation using masked rbms. *International Conference on Artificial Neural Networks*, 2011.

[13] G. E. Hinton. Training products of experts by minimizing contrastive divergence. *Neural Computation*, 14(8):1771–1800, 2002.

[14] Y. Hoffman and E. Ribak. Constrained realizations of Gaussian fields: a simple algorithm. *The Astrophysical Journal*, 380:L5–L8, 1991.

[15] I. Murray and R. Salakhutdinov. Notes on the KL-divergence between a Markov chain and its equilibrium distribution, 2008.

[16] R. M. Neal. Annealed importance sampling. *Statistics and Computing*, 11(2):125–139, 2001.

[17] R. M. Neal. *MCMC using Hamiltonian Dynamics*, pages 113–162. Chapman & Hall/CRC Press, 2011.

[18] R. M. Neal and G. E. Hinton. *A view of the EM algorithm that justifies incremental, sparse, and other variants*, pages 355–368. MIT Press, 1998.

[19] B. A. Olshausen and D. J. Field. Emergence of simple-cell receptive field properties by learning a sparse code for natural images. *Nature*, 381:607–609, 1996.

[20] B. A. Olshausen and D. J. Field. Sparse coding with an overcomplete basis set: A strategy employed by V1? *Vision Research*, 37(23):3311–3325, 1997.

[21] B. A. Olshausen and K. J. Millman. Learning sparse codes with a mixture-of-Gaussians prior. *Advances in Neural Information Processing Systems*, 12, 2000.

[22] G. Papandreou and A. L. Yuille. Gaussian sampling by local perturbations. *Advances in Neural Information Processing Systems*, 23, 2010.

[23] T. Park and G. Casella. The Bayesian lasso. *Journal of the American Statistical Association*, 103(482): 681–686, 2008.

[24] R. Penrose. A generalized inverse for matrices. *Proceedings of the Cambridge Philosophical Society*, 51:406–413, 1955.

[25] G. O. Roberts and R. L. Tweedie. Exponential convergence of Langevin diffusions and their discrete approximations. *Bernoulli*, 2(4):341–363, 1996.

[26] B. Sallans. A hierarchical community of experts. Master's thesis, University of Toronto, 1998.

[27] U. Schmidt, Q. Gao, and S. Roth. A generative perspective on MRFs in low-level vision. *Proceedings of the IEEE Conference on Computer Vision and Pattern Recognition*, 2010.

[28] M. W. Seeger. Bayesian inference and optimal design for the sparse linear model. *Journal of Machine Learning Research*, 9:759–813, 2008.

[29] J. Sohl-Dickstein. Persistent minimum probability flow, 2011.

[30] J. Sohl-Dickstein and B. J. Culpepper. Hamiltonian annealed importance sampling for partition function estimation, 2012.

[31] J. Sohl-Dickstein, P. Battaglino, and M. R. DeWeese. Minimum probability flow learning. *Proceedings of the 28th International Conference on Machine Learning*, 2011.

[32] T. Tieleman. Training restricted Boltzmann machines using approximations to the likelihood gradient. *Proceedings of the 25th International Conference on Machine Learning*, 2008.

[33] J. H. van Hateren and A. van der Schaaf. Independent component filters of natural images compared with simple cells in primary visual cortex. *Proc. of the Royal Society B: Biological Sciences*, 265(1394), 1998.

[34] G. C. G. Wei and M. A. Tanner. A Monte Carlo implementation of the EM algorithm and the poor man's data augmentation algorithms. *Journal of the American Statistical Association*, 85(411):699–704, 1990.

[35] M. Welling, G. Hinton, and S. Osindero. Learning sparse topographic representations with products of Student-t distributions. *Advances in Neural Information Processing Systems*, 15, 2003.

[36] L. Younes. Parametric inference for imperfectly observed Gibbsian fields. *Probability Theory and Related Fields*, 1999.

